# ARTEX: A Self-Organizing Architecture for Classifying Image Regions

**Stephen Grossberg and James R. Williamson**
{steve, jrw}@cns.bu.edu
Center for Adaptive Systems and
Department of Cognitive and Neural Systems
Boston University
677 Beacon Street,
Boston, MA 02215

## Abstract

A self-organizing architecture is developed for image region classification. The system consists of a preprocessor that utilizes multi-scale filtering, competition, cooperation, and diffusion to compute a vector of image boundary and surface properties, notably texture and brightness properties. This vector inputs to a system that incrementally learns noisy multidimensional mappings and their probabilities. The architecture is applied to difficult real-world image classification problems, including classification of synthetic aperture radar and natural texture images, and outperforms a recent state-of-the-art system at classifying natural textures.

## 1 INTRODUCTION

Automatic processing of visual scenes often begins by detecting regions of an image with common values of simple local features, such as texture, and mapping the pattern of feature activation into a predicted region label. We develop a self-organizing neural architecture, called the ARTEX algorithm, for automatically extracting a novel and effective array of such features and mapping them to output region labels. ARTEX is made up of biologically motivated networks, the Boundary Contour System and Feature Contour System (BCS/FCS) networks for visual feature extraction (Cohen & Grossberg, 1984; Grossberg & Mingolla, 1985a, 1985b; Grossberg & Todorović, 1988; Grossberg, Mingolla, & Williamson, 1995), and the Gaussian ARTMAP (GAM) network for classification (Williamson, 1996).

ARTEX is first evaluated on a difficult real-world task, classifying regions of synthetic aperture radar (SAR) images, where it reliably achieves high resolution (single

pixel) classification results, and creates accurate probability maps for its class predictions. ARTEX is then evaluated on classification of natural textures, where it outperforms the texture classification system in Greenspan, Goodman, Chellappa, & Anderson (1994) using comparable preprocessing and training conditions.

## 2  FEATURE EXTRACTION NETWORKS

**Filled-in surface brightness.**  Regions of interest in an image can often be segmented based on first-order differences in pixel intensity. An improvement over raw pixel intensities can be obtained by compensating for variable illumination of the image to yield a local brightness feature. A further improvement over local brightness features can be obtained with a surface brightness feature, which is obtained by smoothing local brightness values when they belong to the same region, while maintaining differences when they belong to different regions. Such a procedure tends to maximize the separability of different regions in brightness space by minimizing within-region variance while maximizing between-region variance.

In Grossberg et al. (1995) a multiple-scale BCS/FCS network was used to process noisy SAR images for use by human operators by normalizing and segmenting the SAR intensity distributions and using these transformed data to fill-in surface representations that smooth over noise while maintaining informative structures. The single-scale BCS/FCS used here employs the middle-scale BCS/FCS used in that study. The BCS/FCS equations and parameters are fully described in Grossberg et al. (1995). The BCS/FCS is herein applied to SAR images that are spatially consolidated to half the size (in each dimension) of the images used in that study, and so is comparable to the large-scale BCS/FCS used there.

**Multiple-scale oriented contrast.**  In addition to surface brightness, another image property that is useful for region segmentation is texture. One popular approach for analyzing texture, for which there is a great deal of supporting biological and computational evidence, decomposes an image, at each image location, into a set of energy measures at different oriented spatial frequencies. This may be done by applying a bank of orientation-selective bandpass filters followed by simple nonlinearities and spatial pooling, to extract multiple-scale oriented contrast features. The early stages of the BCS, which define a Static Oriented Constrast (or SOC) filtering network, carry out these operations, and variants of them have been used in many texture segregation algorithms (Bergen, 1991; Greenspan et al., 1994).

Here, the SOC network produces $K = 4$ oriented contrast features at each of four spatial scales. The first stage of the SOC network is a shunting on-center off-surround network that compensates for variable illumination, normalizes, and computes ratio contrasts in the image. Given an input image, $I$, the output at pixel $(i, j)$ and scale $g$ in the first stage of the SOC network is

$$a_{ij}^g = \frac{I_{ij} - (G^g * I)_{ij} - DE}{D + I_{ij} + (G^g * I)_{ij}}, \qquad (1)$$

where $E = 0.5$, and $G^g$ is a Gaussian kernel defined by

$$G_{ij}^g(p, q) = \frac{1}{2\pi\sigma_g^2} \exp[-((i - p)^2 + (j - q)^2)/2\sigma_g^2], \qquad (2)$$

with $\sigma_g = 2^g$, for the spatial scales $g = 0, 1, 2, 3$. The value of $D$ is determined by the range of pixel intensities in the input image. We use $D = 2000$ for SAR images and $D = 255$ for natural texture images. The next stage obtains a local measure of orientational contrast by convolving the output of (1) with Gabor filters, $H_k^g$, which

are defined at four orientations, and then full-wave rectifying the result:

$$b^g_{ijk} = |(H^g_k * a^g)_{ij}|. \tag{3}$$

The horizontal Gabor filter ($k=0$) is defined by:

$$H^g_{ij0}(p,q) = G^g_{ij}(p,q) \cdot \sin[0.75\pi(j-q)/\sigma_g]. \tag{4}$$

Orientational contrast responses may exhibit high spatial variability. A smooth, reliable measure of orientational contrast is obtained by spatially pooling the responses within the same orientation:

$$c^g_{ijk} = (G^g * b^g_k)_{ij}. \tag{5}$$

Equation (5) yields an *orientationally variant*, or OV, representation of oriented contrast. A further optional stage yields an *orientationally invariant*, or OI, representation by shifting the oriented responses at each scale into a canonical ordering, to yield a common representation for rotated versions of the same texture:

$$d^g_{ijk} = c^g_{ijk'}, \text{ where } k' = [k + \arg\max_{k''} (c^g_{ijk''})] \bmod K. \tag{6}$$

# 3   CLASSIFICATION NETWORK

GAM is a constructive, incremental-learning network which self-organizes internal category nodes that learn a Gaussian mixture model of the M-dimensional input space, as well as mappings to output class labels. Here, mappings are learned from 17-dimensional input vectors (composed of a filled-in brightness feature and 16 oriented contrast features) to a class label representing a shadow, road, grass, or tree region. The $j^{th}$ category's receptive field is parametrized by two M-dimensional vectors: its mean, $\vec{\mu}_j$, and standard deviation, $\vec{\sigma}_j$. A scalar, $n_j$, also represents the node's cumulative credit. Category $j$ is activated only if its *match*, $G_j$, satisfies the match criterion, which is determined by a vigilance parameter, $\rho$. Match is a measure, obtained from the category's unit-height Gaussian distribution, of how close an input, $\vec{x}$, is to the category's mean, relative to its standard deviation:

$$G_j = \exp\left(-\frac{1}{2}\sum_{i=1}^{M}\left(\frac{x_i - \mu_{ji}}{\sigma_{ji}}\right)^2\right). \tag{7}$$

The match criterion is a threshold: the category is activated only if $G_j > \rho$; otherwise, the category is *reset*. The input strength, $g_j$, is determined by

$$g_j = \frac{n_j}{\prod_{i=1}^{M}\sigma_{ji}} G_j \text{ if } G_j > \rho; \quad g_j = 0 \text{ otherwise.} \tag{8}$$

The category's activation, $y_j$, which represents $P(j|\vec{x})$, is obtained by

$$y_j = \frac{g_j}{D + \sum_{l=1}^{N} g_l}, \tag{9}$$

where $N$ is the number of categories and $D$ is a shunting decay term that maintains sensitivity to the input magnitude in the activation level ($D = 0.01$ here).

When category $j$ is first chosen, it learns a permanent mapping to the output class, $k$, associated with the current training sample. All categories that map to the same class prediction belong to the same *ensemble*: $j \in E(k)$. Each time an input is presented, the categories in each ensemble sum their activations to generate a net probability estimate, $z_k$, of the class prediction $k$ that they share:

$$z_k = \sum_{j\in E(k)} y_j. \tag{10}$$

The system prediction, $K$, is determined by the maximum probability estimate,

$$K = \arg\max_k(z_k),\qquad(11)$$

which determines the chosen ensemble. Once the class prediction $K$ is chosen, we obtain the category's "chosen-ensemble" activation, $y_j^*$, which represents $P(j|\vec{x}, K)$:

$$y_j^* = \frac{y_j}{\sum_{l\in E(K)} y_l} \quad\text{if}\quad j \in E(K); \quad y_j^* = 0 \text{ otherwise.}\qquad(12)$$

If $K$ is the correct prediction, then the network resonates and learns; otherwise, match tracking is invoked: $\rho$ is raised to the average match of the chosen ensemble.

$$\rho = \exp\left(-\frac{1}{2}\sum_{j\in E(K)} y_j^* \sum_{i=1}^{M}\left(\frac{x_i - \mu_{ji}}{\sigma_{ji}}\right)^2\right).\qquad(13)$$

In addition, all categories in the chosen ensemble are reset. Equations (8)–(11) are then re-evaluated. Based on the remaining non-reset categories, a new prediction $K$ in (11), and its corresponding ensemble, are chosen. This automatic search cycle continues until the correct prediction is made, or until all committed categories are reset and an uncommitted category is chosen. Upon presentation of the next training sample, $\rho$ is reassigned its baseline value: $\rho = \overline{\rho}$. Here, $\overline{\rho} \approx 0$.

When category $j$ learns, $n_j$ is updated to represent the amount of training data the node has been assigned credit for:

$$n_j := n_j + y_j^*.\qquad(14)$$

The vectors $\vec{\mu}_j$ and $\vec{\sigma}_j$ are then updated to learn the input statistics:

$$\mu_{ji} := (1 - y_j^* n_j^{-1})\mu_{ji} + y_j^* n_j^{-1} x_i,\qquad(15)$$

$$\sigma_{ji} := \sqrt{(1 - y_j^* n_j^{-1})\sigma_{ji}^2 + y_j^* n_j^{-1}(x_i - \mu_{ji})^2},\qquad(16)$$

GAM is initialized with $N = 0$. When a category is first chosen, $N$ is incremented, and the new category, indexed by $J = N$, is initialized with $n_J = 1$, $\vec{\mu} = \vec{x}$, $\sigma_{ji} = \gamma$, and with a permanent mapping to the correct output class. Initializing $\sigma_{ji} = \gamma$ is necessary to make (7) and (8) well-defined. Varying $\gamma$ has a marked effect on learning: as $\gamma$ is raised, learning becomes slower, but fewer categories are created. The input vectors are normalized to have the same standard deviation in each dimension so that $\gamma$ has the same meaning in each dimension.

## 4  SIMULATION RESULTS

**Classifying SAR image regions.** Figure 1 illustrates the classification results obtained on one SAR image after training on the other eight images in the data set. The final classification result (bottom, right) closely resembles the hand-labeled regions (middle, left). The caption summarizes the average results obtained on all nine images. ARTEX learns this problem very quickly, using a small number of self-organized categories, as shown in Figure 2 (left). The best classification result of 84.2% correct is obtained by filling-in the probability estimates from equation (10) within the BCS boundaries, using an FCS diffusion equation as described in Grossberg et al. (1995). These filled-in probability estimates predict the actual classification rates with remarkable accuracy (Figure 2, right).

**Classifying natural textures.** ARTEX performance is now compared to that of a texture analysis system described in Greenspan et al. (1994), which we refer to as the "hybrid system" because it is a hybrid architecture made up of a

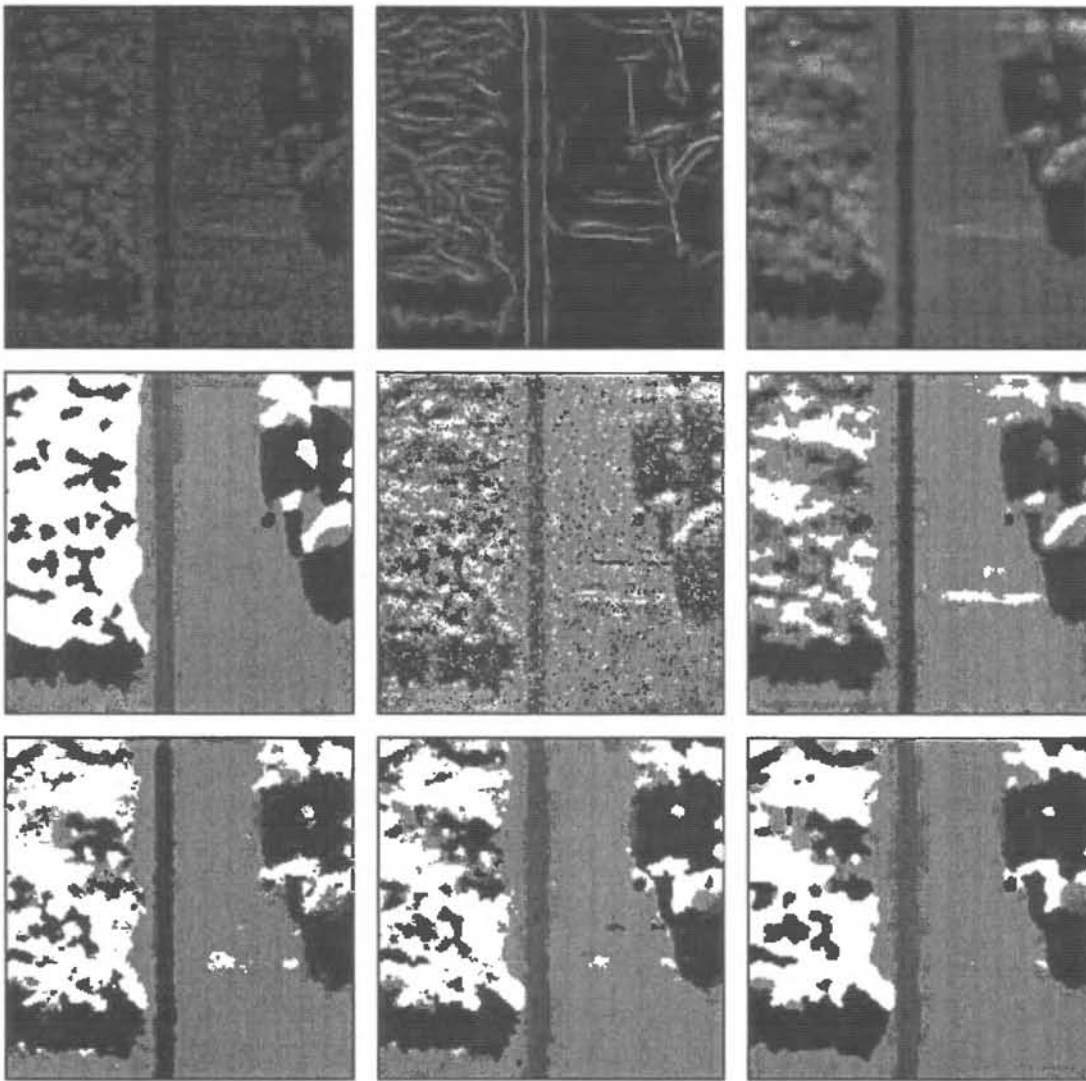

Figure 1: Results are shown on a 180x180 pixel SAR image, which is one of nine images in data set. Top row: Center/surround, first stage output (left); BCS boundaries to FCS filling-in (middle); final BCS/FCS filled-in output (right). Note that BCS accurately localizes region boundaries, and that FCS improves appearance by smoothing intensities within regions while maintaining sharp differences between regions. Middle row: Hand-labeled regions corresponding to shadow, road, grass, trees (left); Gaussian classifier results based on center/surround feature (middle, 59.6% correct), and based on filled-in feature (right, 70.7%). Note that filling-in greatly improves classification by reducing brightness variability within regions. However, the lack of textural information results in errors, such as the misclassification of the vertical road as a shadow region. Bottom row: GAM results ($\gamma = 4$) based on 16 SOC features in addition to the filled-in brightness feature: using the OV representation (left, 81.9%), using the OI representation (middle, 83.2%), and using filled-in OI prediction probabilities (right, 84.2%). With the OV representation (bottom, left), the thin vertical road is misclassified as shadows because there are no thin vertical roads in the training set. With the OI representation, however (bottom, middle), the road is classified correctly because the training set includes thin roads at other orientations. Finally, the classification results are improved by filling-in the prediction probabilities from equation (10) within the BCS boundaries, thereby taking advantage of spatial and structural context (bottom, right).

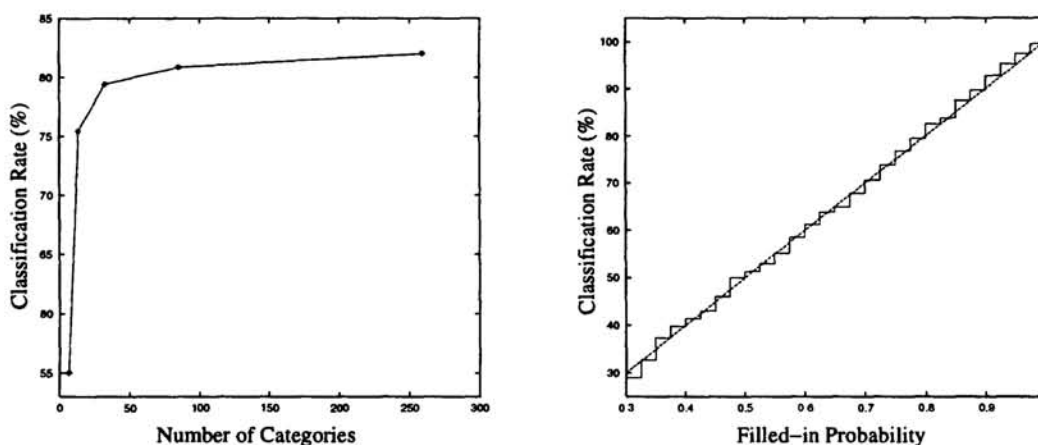

Figure 2: Left: classification rate is plotted as a function of the number of categories after training on different sized subsets of the SAR training data: (left-to-right) 0.01%, 0.1%, 1%, 10%, and 100% of the training set. Right: classification rate is plotted as a function of filled-in probability estimates.

log-Gabor pyramid representation, followed by unsupervised k-means clustering in the feature space, followed by batch learning of mappings from clusters to output classes using a rule-based classifier. The hybrid system uses three pyramid levels and four orientations at each level. Each level of the pyramid is produced via three blurring/decimation steps, resulting in an 8x8 pixel resolution. For a fair comparison, sufficient blurring/decimation was added as a postprocessing step to ARTEX features to yield the same net amount of blurring. Both ARTEX and the hybrid system use an OV representation for these problems because the textures are not rotated. The first task is classification of a library of ten separate structured and unstructured textures after training on different example images. ARTEX obtains better performance, achieving 96.3% correct after 40 training epochs (with $\gamma = 1$, 34 categories) versus 94.3% for the hybrid system. Even after only one training epoch, ARTEX achieves better results (94.9%, 23 categories). The second task (Figure 3) is classification of a five-texture mosaic, which requires discriminating texture boundaries, after training on examples of the five textures, plus an additional texture (sand). ARTEX achieves 93.6% correct after 40 training epochs (33 categories), and produces results which appear to be better than those produced by the hybrid system on a similar problem (see Greenspan et al., 1994, Figure 5).

In summary, the ARTEX system demonstrates the utility of combining BCS texture and FCS brightness measures for image preprocessing. These features may be effectively classified by the GAM network, whose self-calibrating matching and search operations enable it to carry out fast, incremental, distributed learning of recognition categories and their probabilities. BCS boundaries may be further used to constrain the diffusion of these probabilities according to FCS rules to improve prediction probability.

## Acknowledgements

Stephen Grossberg was supported by the Office of Naval Research (ONR N00014-95-1-0409 and ONR N00014-95-1-0657). James Williamson was supported by the Advanced Research Projects Agency (ONR N00014-92-J-4015), the Air Force Office of Scientific Research (AFOSR F49620-92-J-0225 and AFOSR F49620-92-J-0334), the National Science Foundation (NSF IRI-90-00530 and NSF IRI-90-24877), and

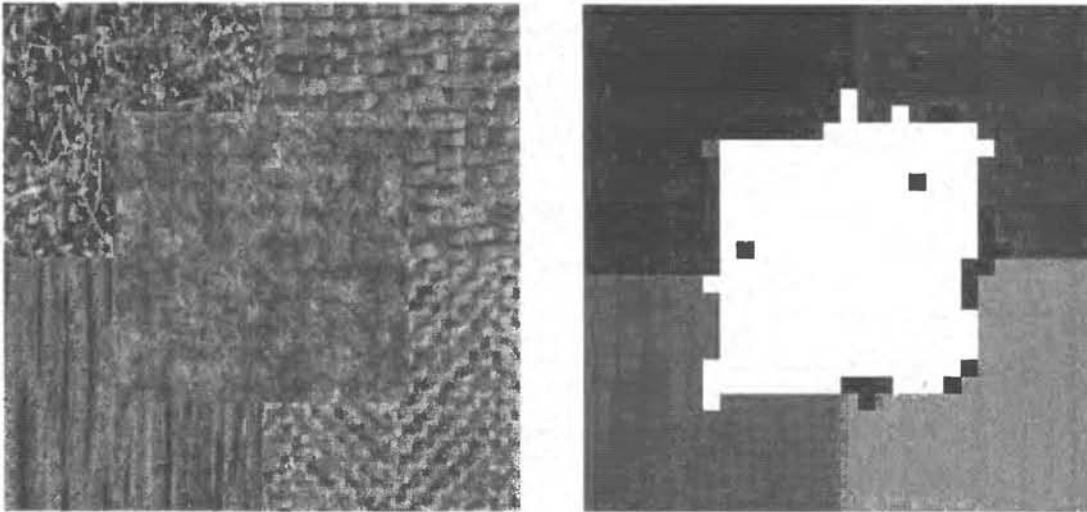

Figure 3: Left: mosaic of five natural textures. Right: ARTEX classification (93.6% correct) after training on examples of five textures and an additional texture (sand).

the Office of Naval Research (ONR N00014-91-J-4100 and ONR N00014-95-1-0409).

# References

Bergen, J.R. "Theories of visual texture perception," in *Spatial Vision*, D. M. Regan Ed. New York: Macmillan, 1991, pp. 114-134.

Cohen, M. & Grossberg, S., (1984). Neural dynamics of brightness perception: Features, boundaries, diffusion, and resonance. *Perception & Psychophysics*, **36**, 428-456.

Greenspan, H., Goodman, R., Chellappa, R., & Anderson, C.H. (1994). Learning texture discrimination rules in a multiresolution system. *IEEE Trans. PAMI*, **16**, 894-901.

Grossberg, S. & Mingolla, E. (1985a). Neural dynamics of form perception: Boundary completion, illusory figures, and neon color spreading. *Psychological Review*, **92**, 173-211.

Grossberg, S. & Mingolla, E. (1985b). Neural dynamics of perceptual grouping: Textures, boundaries, and emergent segmentations. *Perception & Psychophysics*, **38**, 141-171.

Grossberg, S., Mingolla, E., & Williamson, J. (1995). Synthetic aperture radar processing by a multiple scale neural system for boundary and surface representation. *Neural Networks*, **8**, 1005-1028.

Grossberg, S. & Todorović, D. (1988). Neural dynamics of 1-D and 2-D brightness perception: A unified model of classical and recent phenomena. *Perception & Psychophysics*, **43**, 241-277.

Grossberg, S., & Williamson, J.R. (1996). A self-organizing system for classifying complex images: Natural textures and synthetic aperture radar. Technical Report CAS/CNS TR-96-002, Boston, MA: Boston University.

Williamson, J.R. (1996). Gaussian ARTMAP: A neural network for fast incremental learning of noisy multidimensional maps. *Neural Networks*, **9**, 881-897.
